# Exponentiated Gradient Algorithms for Large-margin Structured Classification

**Peter L. Bartlett**
U.C.Berkeley
bartlett@stat.berkeley.edu

**Michael Collins**
MIT CSAIL
mcollins@csail.mit.edu

**Ben Taskar**
Stanford University
btaskar@cs.stanford.edu

**David McAllester**
TTI at Chicago
mcallester@tti-c.org

## Abstract

We consider the problem of *structured classification*, where the task is to predict a label $y$ from an input $x$, and $y$ has meaningful internal structure. Our framework includes supervised training of Markov random fields and weighted context-free grammars as special cases. We describe an algorithm that solves the large-margin optimization problem defined in [12], using an exponential-family (Gibbs distribution) representation of structured objects. The algorithm is efficient—even in cases where the number of labels $y$ is exponential in size—provided that certain expectations under Gibbs distributions can be calculated efficiently. The method for structured labels relies on a more general result, specifically the application of exponentiated gradient updates [7, 8] to quadratic programs.

## 1   Introduction

Structured classification is the problem of predicting $y$ from $x$ in the case where $y$ has meaningful internal structure. For example $x$ might be a word string and $y$ a sequence of part of speech labels, or $x$ might be a Markov random field and $y$ a labeling of $x$, or $x$ might be a word string and $y$ a parse of $x$. In these examples the number of possible labels $y$ is exponential in the size of $x$. This paper presents a training algorithm for a general definition of structured classification covering both Markov random fields and parsing.

We restrict our attention to linear discriminative classification. We assume that pairs $\langle x, y \rangle$ can be embedded in a linear feature space $\Phi(x, y)$, and that a predictive rule is determined by a direction (weight vector) $\mathbf{w}$ in that feature space. In linear discriminative prediction we select the $y$ that has the greatest value for the inner product $\langle \Phi(x, y), \mathbf{w} \rangle$. Linear discrimination has been widely studied in the binary and multiclass setting [6, 4]. However, the case of structured labels has only recently been considered [2, 12, 3, 13]. The structured-label case takes into account the internal structure of $y$ in the assignment of feature vectors, the computation of loss, and the definition and use of margins.

We focus on a formulation where each label $y$ is represented as a set of "parts", or equivalently, as a bit-vector. Moreover, we assume that the feature vector for $y$ and the loss for $y$ are both linear in the individual bits of $y$. This formulation has the advantage that it naturally covers both simple labeling problems, such as part-of-speech tagging, as well as more complex problems such as parsing.

We consider the large-margin optimization problem defined in [12] for selecting the classification direction $\mathbf{w}$ given a training sample. The starting-point for these methods is a

primal problem that has one constraint for each possible labeling $y$; or equivalently a dual problem where each $y$ has an associated dual variable. We give a new training algorithm that relies on an exponential-family (Gibbs distribution) representation of structured objects. The algorithm is efficient—even in cases where the number of labels $y$ is exponential in size—provided that certain expectations under Gibbs distributions can be calculated efficiently. The computation of these expectations appears to be a natural computational problem for structured problems, and has specific polynomial-time dynamic programming algorithms for some important examples: for example, the clique-tree belief propagation algorithm can be used in Markov random fields, and the inside-outside algorithm can be used in the case of weighted context-free grammars.

The optimization method for structured labels relies on a more general result, specifically the application of exponentiated gradient (EG) updates [7, 8] to quadratic programs (QPs). We describe a method for solving QPs based on EG updates, and give bounds on its rate of convergence. The algorithm uses multiplicative updates on dual parameters in the problem. In addition to their application to the structured-labels task, the EG updates lead to simple algorithms for optimizing "conventional" binary or multiclass SVM problems.

**Related work**   [2, 12, 3, 13] consider large-margin methods for Markov random fields and (weighted) context-free grammars. We consider the optimization problem defined in [12]. [12] use a row-generation approach based on Viterbi decoding combined with an SMO optimization method. [5] describe exponentiated gradient algorithms for SVMs, but for binary classification in the "hard-margin" case, without slack variables. We show that the EG-QP algorithm converges significantly faster than the rates shown in [5]. Multiplicative updates for SVMs are also described in [11], but unlike our method, the updates in [11] do not appear to factor in a way that allows algorithms for MRFs and WCFGs based on Gibbs-distribution representations. Our algorithms are related to those for conditional random fields (CRFs) [9]. CRFs define a linear model for structured problems, in a similar way to the models in our work, and also rely on the efficient computation of marginals in the training phase. Finally, see [1] for a longer version of the current paper, which includes more complete derivations and proofs.

## 2   The General Setting

We consider the problem of learning a function $f : \mathcal{X} \to \mathcal{Y}$, where $\mathcal{X}$ is a set and $\mathcal{Y}$ is a countable set. We assume a loss function $L : \mathcal{X} \times \mathcal{Y} \times \mathcal{Y} \to \mathbb{R}^+$. The function $L(x, y, \hat{y})$ measures the loss when $y$ is the true label for $x$, and $\hat{y}$ is a predicted label; typically, $\hat{y}$ is the label proposed by some function $f(x)$. In general we will assume that $L(x, y, \hat{y}) = 0$ for $y = \hat{y}$. Given some distribution over examples $(X, Y)$ in $\mathcal{X} \times \mathcal{Y}$, our aim is to find a function with low expected loss, or risk, $\mathbf{E}L(X, Y, f(X))$.

We consider functions $f$ which take a linear form. First, we assume a fixed function $\mathbf{G}$ which maps an input $x$ to a set of candidates $\mathbf{G}(x)$. For all $x$, we assume that $\mathbf{G}(x) \subseteq \mathcal{Y}$, and that $\mathbf{G}(x)$ is finite. A second component to the model is a feature-vector representation $\Phi : \mathcal{X} \times \mathcal{Y} \to \mathbb{R}^d$. Given a parameter vector $\mathbf{w} \in \mathbb{R}^d$, we consider functions of the form

$$f_{\mathbf{w}}(x) = \arg \max_{y \in \mathbf{G}(x)} \langle \Phi(x, y), \mathbf{w} \rangle.$$

Given $n$ independent training examples $(x_i, y_i)$ with the same distribution as $(X, Y)$, we will formalize a large-margin optimization problem that is a generalization of support vector methods for binary classifiers, and is essentially the same as the formulation in [12]. The optimal parameters are taken to minimize the following regularized empirical risk function:

$$\frac{1}{2}\|\mathbf{w}\|^2 + C \sum_i \left( \max_y \left( L(x_i, y_i, y) - m_{i,y}(\mathbf{w}) \right) \right)_+$$

where $m_{i,y}(\mathbf{w}) = \langle \mathbf{w}, \phi(x_i, y_i) \rangle - \langle \mathbf{w}, \phi(x_i, y) \rangle$ is the "margin" on $(i, y)$ and $(z)_+ = \max\{z, 0\}$. This optimization can be expressed as the primal problem in Figure 1. Following [12], the dual of this problem is also shown in Figure 1. The dual is a quadratic

**Primal problem:**

$$\min_{\mathbf{w},\bar{\epsilon}} \left( \frac{1}{2}\|\mathbf{w}\|^2 + C\sum_i \epsilon_i \right)$$

Subject to the constraints:

$$\forall i, \forall y \in \mathbf{G}(x_i),\ \langle \mathbf{w}, \Phi_{i,y}\rangle \geq L_{i,y} - \epsilon_i$$
$$\forall i,\ \epsilon_i \geq 0$$

**Dual problem:** $\max_{\bar{\alpha}} F(\bar{\alpha})$, where

$$F(\bar{\alpha}) = \Big( C\sum_{i,y} \alpha_{i,y} L_{i,y} -$$
$$\frac{1}{2} C^2 \sum_{i,y}\sum_{j,z} \alpha_{i,y}\alpha_{j,z}\langle \Phi_{i,y}, \Phi_{j,z}\rangle \Big)$$

Subject to the constraints:

$$\forall i,\ \sum_y \alpha_{i,y} = 1;\quad \forall i, y,\ \alpha_{i,y} \geq 0$$

Relationship between optimal values: $\mathbf{w}^* = C\sum_{i,y} \alpha_{i,y}^* \Phi_{i,y}$ where $\mathbf{w}^*$ is the $\arg\min$ of the primal problem, and $\bar{\alpha}^*$ is the $\arg\max$ of the dual problem.

Figure 1: The primal and dual problems. We use the definitions $L_{i,y} = L(x_i, y_i, y)$, and $\Phi_{i,y} = \Phi(x_i, y_i) - \Phi(x_i, y)$. We assume that for all $i$, $L_{i,y} = 0$ for $y = y_i$. The constant $C$ dictates the relative penalty for values of the slack variables $\epsilon_i$ which are greater than 0.

program $F(\bar{\alpha})$ in the dual variables $\alpha_{i,y}$ for all $i = 1\ldots n$, $y \in \mathbf{G}(x_i)$. The dual variables for each example are constrained to form a probability distribution over $\mathcal{Y}$.

## 2.1 Models for structured classification

The problems we are interested in concern structured labels, which have a natural decomposition into "parts". Formally, we assume some countable set of parts, $\mathcal{R}$. We also assume a function $R$ which maps each object $(x, y) \in \mathcal{X} \times \mathcal{Y}$ to a finite subset of $\mathcal{R}$. Thus $R(x, y)$ is the set of parts belonging to a particular object. In addition we assume a feature-vector representation $\phi$ of parts: this is a function $\phi : \mathcal{X} \times \mathcal{R} \rightarrow \mathbb{R}^d$. The feature vector for an object $(x, y)$ is then a sum of the feature vectors for its parts, and we also assume that the loss function $L(x, y, \hat{y})$ decomposes into a sum over parts:

$$\Phi(x,y) = \sum_{r \in R(x,y)} \phi(x,r) \qquad L(x,y,\hat{y}) = \sum_{r \in R(x,\hat{y})} l(x,y,r)$$

Here $\phi(x, r)$ is a "local" feature vector for part $r$ paired with input $x$, and $l(x, y, r)$ is a "local" loss for part $r$ when proposed for the pair $(x, y)$. For convenience we define indicator variables $I(x, y, r)$ which are 1 if $r \in R(x, y)$, 0 otherwise. We also define sets $R(x_i) = \cup_{y \in \mathbf{G}(x_i)} R(x_i, y)$ for all $i = 1 \ldots n$.

**Example 1: Markov Random Fields (MRFs)**  In an MRF the space of labels $\mathbf{G}(x)$, and their underlying structure, can be represented by a graph. The graph $G = (V, E)$ is a collection of vertices $V = \{v_1, v_2, \ldots v_l\}$ and edges $E$. Each vertex $v_i \in V$ has a set of possible labels, $\mathcal{Y}_i$. The set $\mathbf{G}(x)$ is then defined as $\mathcal{Y}_1 \times \mathcal{Y}_2 \ldots \times \mathcal{Y}_l$. Each clique in the graph has a set of possible *configurations*: for example, if a particular clique contains vertices $\{v_3, v_5, v_6\}$, the set of possible configurations of this clique is $\mathcal{Y}_3 \times \mathcal{Y}_5 \times \mathcal{Y}_6$. We define $\mathcal{C}$ to be the set of cliques in the graph, and for any $c \in \mathcal{C}$ we define $\mathcal{Y}(c)$ to be the set of possible configurations for that clique. We decompose each $y \in \mathbf{G}(x)$ into a set of parts, by defining $R(x, y) = \{(c, a) \in \mathcal{R} : c \in \mathcal{C}, a \in \mathcal{Y}(c), (c, a) \text{ is consistent with y}\}$. The feature vector representation $\phi(x, c, a)$ for each part can essentially track any characteristics of the assignment $a$ for clique $c$, together with any features of the input $x$. A number of choices for the loss function $l(x, y, (c, a))$ are possible. For example, consider the Hamming loss used in [12], defined as $L(x, y, \hat{y}) = \sum_i I_{y_i \neq \hat{y}_i}$. To achieve this, first assign each vertex $v_i$ to a single one of the cliques in which it appears. Second, define $l(x, y, (c, a))$ to be the number of labels in the assignment $(c, a)$ which are both incorrect and correspond to vertices which have been assigned to the clique $c$ (note that assigning each vertex to a single clique avoids "double counting" of label errors).

**Example 2: Weighted Context-Free Grammars (WCFGs).**  In this example $x$ is an input string, and $y$ is a "parse tree" for that string, i.e., a left-most derivation for $x$ under some context-free grammar. The set $\mathbf{G}(x)$ is the set of all left-most derivations for $x$

**Inputs:** A learning rate $\eta$.
**Data structures:** A vector $\bar{\theta}$ of variables, $\theta_{i,r}, \forall i, \forall r \in R(x_i)$.

**Definitions:** $\alpha_{i,y}(\bar{\theta}) = \exp(\sum_{r \in R(x_i,y)} \theta_{i,r})/Z_i$ where $Z_i$ is a normalization term.

**Algorithm:**
- Choose initial values $\bar{\theta}^1$ for the $\theta_{i,r}$ variables (these values can be arbitrary).
- For $t = 1 \ldots T + 1$:
    - For $i = 1 \ldots n, r \in R(x_i)$, calculate $\mu_{i,r}^t = \sum_y \alpha_{i,y}(\bar{\theta}^t) I(x_i, y, r)$.
    - Set $\mathbf{w}^t = C \left( \sum_{i,r \in R(x_i,y_i)} \phi_{i,r} - \sum_{i,r \in R(x_i)} \mu_{i,r}^t \phi_{i,r} \right)$
    - For $i = 1 \ldots n, r \in R(x_i)$, calculate updates $\theta_{i,r}^{t+1} = \theta_{i,r}^t + \eta C \left( l_{i,r} + \langle \mathbf{w}^t, \phi_{i,r} \rangle \right)$

**Output:** Parameter values $\mathbf{w}^{T+1}$

Figure 2: The EG algorithm for structured problems. We use $\phi_{i,r} = \phi(x_i, r)$ and $l_{i,r} = l(x_i, y_i, r)$.

under the grammar. For convenience, we restrict the grammar to be in Chomsky-normal form, where all rules in the grammar are of the form $\langle A \to B \ C \rangle$ or $\langle A \to a \rangle$, where $A, B, C$ are non-terminal symbols, and $a$ is some terminal symbol. We take a part $r$ to be a CF-rule-tuple $\langle A \to B \ C, s, m, e \rangle$. Under this representation $A$ spans words $s \ldots e$ inclusive in $x$; $B$ spans words $s \ldots m$; and $C$ spans words $(m+1) \ldots e$. The function $R(x, y)$ maps a derivation $y$ to the set of parts which it includes. In WCFGs $\phi(x, r)$ can be any function mapping a rule production and its position in the sentence $x$, to a feature vector. One example of a loss function would be to define $l(x, y, r)$ to be 1 only if $r$'s non-terminal $A$ is not seen spanning words $s \ldots e$ in the derivation $y$. This would lead to $L(x, y, \hat{y})$ tracking the number of "constituent errors" in $\hat{y}$, where a constituent is a (non-terminal, start-point, end-point) tuple such as $(A, s, e)$.

## 3 EG updates for structured objects

We now consider an algorithm for computing $\bar{\alpha}^* = \arg \max_{\bar{\alpha} \in \Delta} F(\bar{\alpha})$, where $F(\bar{\alpha})$ is the dual form of the maximum margin problem, as in Figure 1. In particular, we are interested in the optimal values of the primal form parameters, which are related to $\bar{\alpha}^*$ by $\mathbf{w}^* = C \sum_{i,y} \alpha_{i,y}^* \Phi_{i,y}$. A key problem is that in many of our examples, the number of dual variables $\alpha_{i,y}$ precludes dealing with these variables directly. For example, in the MRF case or the WCFG cases, the set $\mathbf{G}(x)$ is exponential in size, and the number of dual variables $\alpha_{i,y}$ is therefore also exponential.

We describe an algorithm that is efficient for certain examples of structured objects such as MRFs or WCFGs. Instead of representing the $\alpha_{i,y}$ variables explicitly, we will instead manipulate a vector $\bar{\theta}$ of variables $\theta_{i,r}$ for $i = 1 \ldots n, r \in R(x_i)$. Thus we have one of these "mini-dual" variables for each part seen in the training data. Each of the variables $\theta_{i,r}$ can take any value in the reals. We now define the dual variables $\alpha_{i,y}$ as a function of the vector $\bar{\theta}$, which takes the form of a Gibbs distribution:

$$\alpha_{i,y}(\bar{\theta}) = \frac{\exp(\sum_{r \in R(x_i,y)} \theta_{i,r})}{\sum_{y'} \exp(\sum_{r \in R(x_i,y')} \theta_{i,r})} .$$

Figure 2 shows an algorithm for maximizing $F(\bar{\alpha})$. The algorithm defines a sequence of values $\bar{\theta}^1, \bar{\theta}^2, \ldots$. In the next section we prove that the sequence $F(\bar{\alpha}(\bar{\theta}^1)), F(\bar{\alpha}(\bar{\theta}^2)), \ldots$ converges to $\max_\alpha F(\bar{\alpha})$. The algorithm can be implemented efficiently, independently of the dimensionality of $\bar{\alpha}$, provided that there is an efficient algorithm for computing *marginal* terms $\mu_{i,r} = \sum_{i,y} \alpha_{i,y}(\bar{\theta}) I(x_i, y, r)$ for all $i = 1 \ldots n, r \in R(x_i)$, and all $\bar{\theta}$. A key property is that the primal parameters $\mathbf{w} = C \sum_{i,y} \alpha_{i,y}(\bar{\theta}) \Phi_{i,y} = C \sum_i \Phi(x_i, y_i) -$

$C \sum_{i,y} \alpha_{i,y}(\bar{\theta}) \Phi(x_i, y)$ can be expressed in terms of the marginal terms, because:

$$\sum_{i,y} \alpha_{i,y}(\bar{\theta}) \Phi(x_i, y) = \sum_{i,y} \alpha_{i,y}(\bar{\theta}) \sum_{r \in R(x_i, y)} \phi(x_i, r) = \sum_{i, r \in R(x_i)} \mu_{i,r} \phi(x_i, r)$$

and hence $\mathbf{w} = C \sum_i \Phi(x_i, y_i) - C \sum_{i, r \in R(x_i)} \mu_{i,r} \phi(x_i, r)$. The $\mu_{i,r}$ values can be calculated for MRFs and WCFGs in many cases, using standard algorithms. For example, in the WCFG case, the inside-outside algorithm can be used, provided that each part $r$ is a context-free rule production, as described in Example 2 above. In the MRF case, the $\mu_{i,r}$ values can be calculated efficiently if the tree-width of the underlying graph is small.

Note that the main storage requirements of the algorithm in Figure 2 concern the vector $\bar{\theta}$. This is a vector which has as many components as there are parts in the training set. In practice, the number of parts in the training data can become extremely large. Fortunately, an alternative, "primal form" algorithm is possible. Rather than explicitly storing the $\theta_{i,r}$ variables, we can store a vector $\mathbf{z}^t$ of the same dimensionality as $\mathbf{w}^t$. The $\theta_{i,r}$ values can be computed from $\mathbf{z}^t$. More explicitly, the main body of the algorithm in Figure 2 can be replaced with the following:

- Set $\mathbf{z}^1$ to some initial value. For $t = 1 \ldots T + 1$:
  - Set $\mathbf{w}^t = 0$
  - For $i = 1 \ldots n$: Compute $\mu_{i,r}^t$ for $r \in R(x_i)$, using $\theta_{i,r}^t = \eta C((t-1)l_{i,r} + \langle \mathbf{z}^t, \phi_{i,r} \rangle)$;

  $$\text{Set } \mathbf{w}^t = \mathbf{w}^t + C \left( \sum_{r \in R(x_i, y_i)} \phi_{i,r} - \sum_{r \in R(x_i)} \mu_{i,r}^t \phi_{i,r} \right)$$

  - Set $\mathbf{z}^{t+1} = \mathbf{z}^t + \mathbf{w}^t$

It can be verified that if $\forall i, r, \ \theta_{i,r}^1 = \eta C \langle \phi_{i,r}, \mathbf{z}^1 \rangle$, then this alternative algorithm defines the same sequence of (implicit) $\theta_{i,r}^t$ values, and (explicit) $\mathbf{w}^t$ values, as the original algorithm. In the next section we show that the original algorithm converges for any choice of initial values $\bar{\theta}^1$, so this restriction on $\theta_{i,r}^1$ should not be significant.

## 4  Exponentiated gradient (EG) updates for quadratic programs

We now prove convergence properties of the algorithm in Figure 2. We show that it is an instantiation of a general algorithm for optimizing quadratic programs (QPs), which relies on Exponentiated Gradient (EG) updates [7, 8]. In the general problem we assume a positive semi-definite matrix $\mathbf{A} \in \mathbb{R}^{m \times m}$, and a vector $\mathbf{b} \in \mathbb{R}^m$, specifying a loss function $Q(\bar{\alpha}) = \mathbf{b}' \bar{\alpha} + \frac{1}{2} \bar{\alpha}' \mathbf{A} \bar{\alpha}$. Here $\bar{\alpha}$ is an $m$-dimensional vector of reals. We assume that $\bar{\alpha}$ is formed by the concatenation of $n$ vectors $\bar{\alpha}_i \in \mathbb{R}^{m_i}$ for $i = 1 \ldots n$, where $\sum_i m_i = m$. We assume that each $\bar{\alpha}_i$ lies in a simplex of dimension $m_i$, so that the feasible set is

$$\Delta = \{\bar{\alpha} : \bar{\alpha} \in \mathbb{R}^m; \text{for } i = 1 \ldots n, \sum_{j=1}^{m_i} \alpha_{i,j} = 1; \text{for all } i, j, \alpha_{i,j} \geq 0\}. \tag{1}$$

Our aim is to find $\arg \min_{\bar{\alpha} \in \Delta} Q(\bar{\alpha})$. Figure 3 gives an algorithm—the "EG-QP" algorithm—for finding the minimum. In the next section we give a proof of its convergence properties.

The EG-QP algorithm can be used to find the minimum of $-F(\bar{\alpha})$, and hence the maximum of the dual objective $F(\bar{\alpha})$. We justify the algorithm in Figure 2 by showing that it is equivalent to minimization of $-F(\bar{\alpha})$ using the EG-QP algorithm. We give the following theorem:

**Theorem 1** *Define* $F(\bar{\alpha}) = C \sum_{i,y} \alpha_{i,y} L_{i,y} - \frac{1}{2} C^2 \sum_{i,y} \sum_{j,z} \alpha_{i,y} \alpha_{j,z} \langle \Phi_{i,y}, \Phi_{j,z} \rangle$, *and assume as in section 2 that* $L_{i,y} = \sum_{r \in R(x_i, y)} l(x_i, y, r)$ *and* $\Phi(x_i, y) = \sum_{r \in R(x_i, y)} \phi(x_i, r)$. *Consider the sequence* $\bar{\alpha}(\bar{\theta}^1) \ldots \bar{\alpha}(\bar{\theta}^{T+1})$ *defined by the algorithm in Figure 2, and the sequence* $\bar{\alpha}^1 \ldots \bar{\alpha}^{T+1}$ *defined by the EG-QP algorithm when applied to* $Q(\bar{\alpha}) = -F(\bar{\alpha})$. *Then under the assumption that* $\bar{\alpha}(\bar{\theta}^1) = \bar{\alpha}^1$, *it follows that* $\bar{\alpha}(\bar{\theta}^t) = \bar{\alpha}^t$ *for* $t = 1 \ldots (T+1)$.

Figure 3: The EG-QP algorithm for quadratic programs.

**Proof.** We can write $F(\bar{\alpha}) = C \sum_{i,y} \alpha_{i,y} L_{i,y} - \frac{1}{2}C^2 \| \sum_i \Phi(x_i, y_i) - \sum_{i,y} \alpha_{i,y}\Phi(x_i, y)\|^2$. It follows that $\frac{\partial F(\bar{\alpha}^t)}{\partial \alpha_{i,y}} = CL_{i,y} + C\langle\Phi(x_i, y), \mathbf{w}^t\rangle = C\sum_{r \in R(x_i,y)} \left(l_{i,r} + \langle\phi_{i,r}, \mathbf{w}^t\rangle\right)$ where as before $\mathbf{w}^t = C(\sum_i \Phi(x_i, y_i) - \sum_{i,y}\alpha_{i,y}^t\Phi(x_i, y))$. The rest of the proof proceeds by induction; due to space constraints we give a sketch of the proof here. The idea is to show that $\bar{\alpha}(\bar{\theta}^{t+1}) = \bar{\alpha}^{t+1}$ under the inductive hypothesis that $\bar{\alpha}(\bar{\theta}^t) = \bar{\alpha}^t$. This follows immediately from the definitions of the mappings $\bar{\alpha}(\bar{\theta}^t) \to \bar{\alpha}(\bar{\theta}^{t+1})$ and $\bar{\alpha}^t \to \bar{\alpha}^{t+1}$ in the two algorithms, together with the identities $s_{i,y}^t = -\frac{\partial F(\bar{\alpha}^t)}{\partial \alpha_{i,y}} = -C\sum_{r \in R(x_i,y)} \left(l_{i,r} + \langle\phi_{i,r}, \mathbf{w}^t\rangle\right)$ and $\theta_{i,r}^{t+1} - \theta_{i,r}^t = \eta C \left(l_{i,r} + \langle\phi_{i,r}, \mathbf{w}^t\rangle\right)$.

### 4.1 Convergence of the exponentiated gradient QP algorithm

The following theorem shows how the optimization algorithm converges to an optimal solution. The theorem compares the value of the objective function for the algorithm's vector $\bar{\alpha}^t$ to the value for a comparison vector $u \in \Delta$. (For example, consider $u$ as the solution of the QP.) The convergence result is in terms of several properties of the algorithm and the comparison vector $u$. The distance between $u$ and $\bar{\alpha}_1$ is measured using the Kullback-Liebler (KL) divergence. Recall that the KL-divergence between two probability vectors $\bar{u}, \bar{v}$ is defined as $D(\bar{u}, \bar{v}) = \sum_i u_i \log \frac{u_i}{v_i}$. For sequences of probability vectors, $\bar{u} \in \Delta$ with $\bar{u} = (\bar{u}_1, \ldots, \bar{u}_n)$ and $\bar{u}_i = (u_{i,1}, \ldots, u_{i,m_i})$, we can define a divergence as the sum of KL-divergences: for $\bar{u}, \bar{v} \in \Delta$, $\bar{D}(\bar{u}, \bar{v}) = \sum_{i=1}^n D(\bar{u}_i, \bar{v}_i)$. Two other key parameters are $\lambda$, the largest eigenvalue of the positive semidefinite symmetric matrix $A$, and

$$B = \max_{\bar{\alpha} \in \Delta} \left( \max_i \left(\nabla Q(\bar{\alpha})\right)_i - \min_i \left(\nabla Q(\bar{\alpha})\right)_i \right) \leq 2 \left( n \max_{ij} |A_{ij}| + \max_i |b_i| \right).$$

**Theorem 2** *For all $\bar{u} \in \Delta$,*

$$\frac{1}{T}\sum_{t=1}^T Q(\bar{\alpha}^t) \leq Q(\bar{u}) + \frac{\bar{D}(\bar{u}, \bar{\alpha}^1)}{\eta T} + \frac{e^{\eta B} - 1 - \eta B}{\eta^2 B^2 \left(1 - \eta(B + \lambda)e^{\eta B}\right)} \frac{Q(\bar{\alpha}^1) - Q(\bar{\alpha}^{T+1})}{T}.$$

*Choosing $\eta = 0.4/(B + \lambda)$ ensures that*

$$Q\left(\bar{\alpha}^{T+1}\right) \leq \frac{1}{T}\sum_{t=1}^T Q(\bar{\alpha}^t) \leq Q(\bar{u}) + 2.5(B + \lambda)\frac{\bar{D}(\bar{u}, \bar{\alpha}^1)}{T} + 1.5\frac{Q(\bar{\alpha}^1) - Q(\bar{\alpha}^{T+1})}{T}.$$

The first lemma we require is due to Kivinen and Warmuth [8].

**Lemma 1** *For any $\bar{u} \in \Delta$,* $\quad \eta Q(\bar{\alpha}^t) - \eta Q(\bar{u}) \leq \bar{D}(\bar{u}, \bar{\alpha}^t) - \bar{D}(\bar{u}, \bar{\alpha}^{t+1}) + \bar{D}(\bar{\alpha}^t, \bar{\alpha}^{t+1})$

We focus on the third term. Define $\nabla_{(i)}Q(\bar{\alpha})$ as the segment of the gradient vector corresponding to the component $\bar{\alpha}_i$ of $\bar{\alpha}$, and define the random variable $X_{i,t}$, satisfying $\Pr\left(X_{i,t} = -\left(\nabla_{(i)}Q(\bar{\alpha}^t)\right)_j\right) = \alpha_{i,j}$.

**Lemma 2** $\bar{D}(\bar{\alpha}^t, \bar{\alpha}^{t+1}) = \sum_{i=1}^n \log \mathbf{E}\left[e^{\eta(X_{i,t} - \mathbf{E}X_{i,t})}\right] \le \left(\frac{e^{\eta B} - 1 - \eta B}{B^2}\right) \sum_{i=1}^n \mathrm{var}(X_{i,t}).$

**Proof.**
$$\bar{D}(\bar{\alpha}^t, \bar{\alpha}^{t+1}) = \sum_{i=1}^n \sum_j \alpha_{ij}^t \log \frac{\alpha_{ij}^t}{\alpha_{ij}^{t+1}}$$
$$= \sum_{i=1}^n \sum_j \alpha_{ij}^t \left(\log\left(\sum_k \alpha_{ik}^t \exp(-\eta \nabla_{i,k})\right) + \eta \nabla_{i,j}\right)$$
$$= \sum_{i=1}^n \log\left(\sum_k \alpha_{ik}^t \exp\left(-\eta \nabla_{i,k} + \eta \bar{\alpha}_i^t \cdot \nabla_i\right)\right)$$
$$= \sum_{i=1}^n \log\left(\mathbf{E}\left[e^{\eta(X_{i,t} - \mathbf{E}X_{i,t})}\right]\right) \le \frac{e^{\eta B} - 1 - \eta B}{B^2} \sum_{i=1}^n \mathrm{var}(X_{i,t}).$$

This last inequality is at the heart of the proof of Bernstein's inequality; e.g., see [10].

The second part of the proof of the theorem involves bounding this variance in terms of the loss. The following lemma relies on the fact that this variance is, to first order, the decrease in the quadratic loss, and that the second order term in the Taylor series expansion of the loss is small compared to the variance, provided the steps are not too large. The lemma and its proof require several definitions. For any $d$, let $\sigma : \mathbb{R}^d \to (0,1)^d$ be the softmax function, $\sigma(\bar{\theta})_i = \exp(\theta_i) / \sum_{j=1}^d \exp(\theta_j)$, for $\bar{\theta} \in \mathbb{R}^d$. We shall work in the exponential parameter space: let $\bar{\theta}^t$ be the exponential parameters at step $t$, so that the updates are $\bar{\theta}^{t+1} = \bar{\theta}^t - \eta \nabla Q(\bar{\alpha}^t)$, and the QP variables satisfy $\bar{\alpha}_i^t = \sigma(\bar{\theta}_i^t)$. Define the random variables $X_{i,t,\bar{\theta}}$, satisfying $\Pr\left(X_{i,t,\bar{\theta}} = -\left(\nabla_{(i)}Q(\bar{\alpha}^t)\right)_j\right) = \left(\sigma(\bar{\theta}_i)\right)_j$. This takes the same values as $X_{i,t}$, but its distribution is given by a different exponential parameter ($\bar{\theta}_i$ instead of $\bar{\theta}_i^t$). Define $\left[\bar{\theta}^t, \bar{\theta}^{t+1}\right] = \left\{a\bar{\theta}^t + (1-a)\bar{\theta}^{t+1} : a \in [0,1]\right\}$.

**Lemma 3** *For some $\bar{\theta} \in [\bar{\theta}^t, \bar{\theta}^{t+1}]$,*

$$\eta \sum_{i=1}^n \mathrm{var}(X_{i,t}) - \eta^2 (B + \lambda) \sum_{i=1}^n \mathrm{var}(X_{i,t,\bar{\theta}}) \le Q(\bar{\alpha}^t) - Q(\bar{\alpha}^{t+1}),$$

*but for all $\bar{\theta} \in [\bar{\theta}^t, \bar{\theta}^{t+1}]$, $\mathrm{var}(X_{i,t,\bar{\theta}}) \le e^{\eta B} \mathrm{var}(X_{i,t})$. Hence,*

$$\sum_{i=1}^n \mathrm{var}(X_{i,t}) \le \frac{1}{\eta \left(1 - \eta(B + \lambda)e^{\eta B}\right)} \left(Q(\bar{\alpha}^t) - Q(\bar{\alpha}^{t+1})\right).$$

*Thus, for $\eta < 0.567/(B + \lambda)$, $Q(\bar{\alpha}^t)$ is non-increasing in $t$.*

The proof is in [1]. Theorem 2 follows from an easy calculation.

## 5  Experiments

We compared an online[1] version of the Exponentiated Gradient algorithm with the factored Sequential Minimal Optimization (SMO) algorithm in [12] on a sequence segmentation task. We selected the first 1000 sentences (12K words) from the CoNLL-2003 named entity recognition challenge data set for our experiment. The goal is to extract (multi-word) entity names of people, organizations, locations and miscellaneous entities. Each word is labelled by 9 possible tags (beginning of one of the four entity types, continuation of one of the types, or not-an-entity). We trained a first-order Markov chain over the tags,

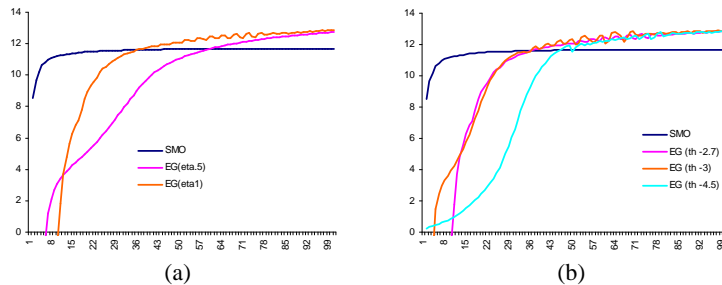

Figure 4: Number of iterations over training set vs. dual objective for the SMO and EG algorithms. (a) Comparison with different $\eta$ values; (b) Comparison with $\eta = 1$ and different initial $\theta$ values.

where our cliques are just the nodes for the tag of each word and edges between tags of consecutive words. The feature vector for each node assignment consists of the word itself, its capitalization and morphological features, etc., as well as the previous and consecutive words and their features. Likewise, the feature vector for each edge assignment consists of the two words and their features as well as surrounding words.

Figure 4 shows the growth of the dual objective function after each pass through the data for SMO and EG, for several settings of the learning rate $\eta$ and the initial setting of the $\theta$ parameters. Note that SMO starts up very quickly but slows down in a suboptimal region, while EG lags at the start, but overtakes SMO and achieves a larger than 10% increase in the value of the objective. These preliminary results suggest that a hybrid algorithm could get the benefits of both, by starting out with several SMO updates and then switching to EG. The key issue is to switch from the marginal $\mu$ representation SMO maintains to the Gibbs $\theta$ representation that EG uses. We can find $\theta$ that produces $\mu$ by first computing conditional "probabilities" that correspond to our marginals (e.g. dividing edge marginals by node marginals in this case) and then letting $\theta$'s be the logs of the conditional probabilities.

## Footnotes

[1]In the online algorithm we calculate marginal terms, and updates to the $\mathbf{w}^t$ parameters, one training example at a time. As yet we do not have convergence bounds for this method, but we have found that it works well in practice.

# References

[1] *Long version of this paper*. Available at `http://www.ai.mit.edu/people/mcollins`.

[2] Y. Altun, I. Tsochantaridis, and T. Hofmann. Hidden markov support vector machines. In *ICML*, 2003.

[3] Michael Collins. Parameter estimation for statistical parsing models: Theory and practice of distribution-free methods. In Harry Bunt, John Carroll, and Giorgio Satta, editors, *New Developments in Parsing Technology*. Kluwer, 2004.

[4] K. Crammer and Y. Singer. On the algorithmic implementation of multiclass kernel-based vector machines. *Journal of Machine Learning Research*, 2(5):265–292, 2001.

[5] N. Cristianini, C. Campbell, and J. Shawe-Taylor. Multiplicative updatings for support-vector learning. Technical report, NeuroCOLT2, 1998.

[6] N. Cristianini and J. Shawe-Taylor. *An Introduction to Support Vector Machines and Other Kernel-Based Learning Methods*. Cambridge University Press, 2000.

[7] J. Kivinen and M. Warmuth. Exponentiated gradient versus gradient descent for linear predictors. *Information and Computation*, 132(1):1–63, 1997.

[8] J. Kivinen and M. Warmuth. Relative loss bounds for multidimensional regression problems. *Journal of Machine Learning Research*, 45(3):301–329, 2001.

[9] John Lafferty, Andrew McCallum, and Fernando Pereira. Conditional random fields: Probabilistic models for segmenting and labeling sequence data. In *Proceedings of ICML-01*, 2001.

[10] D. Pollard. *Convergence of Stochastic Processes*. Springer-Verlag, 1984.

[11] F. Sha, L. Saul, and D. Lee. Multiplicative updates for large margin classifiers. In *COLT*, 2003.

[12] B. Taskar, C. Guestrin, and D. Koller. Max margin Markov networks. In *NIPS*, 2003.

[13] I. Tsochantaridis, T. Hofmann, T. Joachims, and Y. Altun. Support vector machine learning for interdependent and structured output spaces. ICML, 2004 (To appear).
